# Exploration-Exploitation Tradeoffs for Experts Algorithms in Reactive Environments

**Daniela Pucci de Farias**
Department of Mechanical Engineering
Massachusetts Institute of Technology
Cambridge, MA 02139
pucci@mit.edu

**Nimrod Megiddo**
IBM Almaden Research Center
650 Harry Road, K53-B2
San Jose, CA 95120
megiddo@almaden.ibm.com

## Abstract

A reactive environment is one that responds to the actions of an agent rather than evolving obliviously. In reactive environments, experts algorithms must balance exploration and exploitation of experts more carefully than in oblivious ones. In addition, a more subtle definition of a learnable value of an expert is required. A general exploration-exploitation experts method is presented along with a proper definition of value. The method is shown to asymptotically perform as well as the best available expert. Several variants are analyzed from the viewpoint of the exploration-exploitation tradeoff, including explore-then-exploit, polynomially vanishing exploration, constant-frequency exploration, and constant-size exploration phases. Complexity and performance bounds are proven.

## 1  Introduction

Real-world environments require agents to choose actions sequentially. For example, a driver has to choose everyday a route from one point to another, based on past experience and perhaps some current information. In another example, an airline company has to set prices dynamically, also based on past experience and current information. One important difference between these two examples is that the effect of the driver's decision on the future traffic patterns is negligible, whereas prices set by one airline can affect future market prices significantly. In this sense the decisions of the airlines are made in a reactive environment, whereas the driver performs in a non-reactive one. For this reason, the driver's problem is essentially a problem of prediction while the airline's problem has an additional element of control.

In the decision problems we consider, an agent has to repeatedly choose currently feasible actions. The agent then observes a reward, which depends both on the chosen action and the current state of the environment. The state of the environment may depend both on the agent's past choices and on choices made by the environment independent of the agent's current choice. There are various known approaches to sequential decision making under uncertainty. In this paper we focus on the so-called experts algorithm approach. An "expert" (or "oracle") is simply a particular strategy recommending actions based on the past history of the process. An experts algorithm is a method that combines the recommendations of several given "experts" (or "oracles") into another strategy of choosing actions (e.g., [4, 1, 3]).

Many learning algorithms can be interpreted as "exploration-exploitation" methods. Roughly speaking, such algorithms blend choices of exploration, aimed at acquiring knowledge, and exploitation that capitalizes on gained knowledge to accumulate rewards. In particular, some experts algorithms can be interpreted as blending the testing of all experts and following those experts that observed to be more rewarding. Our previous paper [2] presented a specific exploration-exploitation experts algorithm. The reader is referred to [2] for more definitions, examples and discussion. That algorithm was designed especially for learning in reactive environments. The difference between our algorithm and previous experts algorithms is that our algorithm tests each expert for multiple consecutive stages of the decision process, in order to acquire knowledge about how the environment reacts to the expert. We pointed out that the "Minimum Regret" criterion often used for evaluating experts algorithms was not suitable for reactive environments, since it ignored the possibility that different experts may induce different states of the environment. The previous paper, however, did not attempt to optimize the exploration-exploitation tradeoff. It rather focused on one particular possibility, which was shown to perform in the long-run as well as the best expert.

In this paper, we present a more general exploration-exploitation experts method and provide results about the convergence of several of its variants. We develop performance guarantees showing that the method achieves average payoff comparable to that achieved by the best expert. We characterize convergence rates that hold both in expected value and with high probability. We also introduce a definition for the long-term value of an expert, which captures the reactions of the environment to the expert's actions, as well as the fact that any learning algorithm commits mistakes. Finally, we characterize how fast the method learns the value of each expert. An important aspect of our results is that they provide an explicit characterization of the tradeoff between exploration and exploitation.

The paper is organized as follows. The method is described in section 2. Convergence rates based on actual expert performance are presented in section 3. In section 4, we define the experts' long-rum values, whereas in section 5 we address the question of how fast the method learns the values of the experts. Finally, in section 6 we analyze various explorations schemes. These results assume that the number of stages.

## 2 The Exploration-Exploitation Method

The problem we consider in this paper can be described as follows. At times $t = 1, 2, \ldots$, an agent has to choose actions $a_t \in \mathcal{A}$. At the same times the environment also "chooses" $b_t \in \mathcal{B}$, and then the agent receives a reward $R(a_t, b_t)$. The choices of the environment may depend on various factors, including the past choices of the agent.

As in the particular algorithm of [2], the general method follows chosen experts for multiple stages rather than picking a different expert each time. A maximal set of consecutive stages during which the same expert is followed is called a *phase*. Phase numbers are denoted by $i$, The number of phases during which expert $e$ has been followed is denoted by $N_e$, the total number of stages during which expert $e$ has been followed is denoted by $S_e$, and the average payoff from phases in which expert $e$ has been followed is denoted by $M_e$. The general method is stated as follows.

- **Exploration.** An exploration phase consists of picking a random expert $e$ (i.e., from the uniform distribution over $\{1, \ldots, r\}$), and following $e$'s recommendations for a certain number of stages depending on the variant of the method.

- **Exploitation.** An exploitation phase consists of picking an expert $e$ with maximum $M_e$, breaking ties at random, and following $e$'s recommendations for a certain number of stages depending on the variant of the method.

**A general Exploration-Exploitation Experts Method:**

1. Initialize $M_e = N_e = S_e = 0$ $(e = 1, \ldots, r)$ and $i = 1$.

2. With probability $p_i$, perform an exploration phase, and with probability $1 - p_i$ perform an exploitation phase; denote by $e$ the expert chosen to be followed and by $n$ the number of stages chosen for the current phase.

3. Follow expert $e$'s instructions for the next $n$ stages. Increment $N_e = N_e + 1$ and update $S_e = S_e + n$. Denote by $\tilde{R}$ the average payoff accumulated during the current phase of $n$ stages and update

$$M_e = M_e + \frac{n}{S_e}(\tilde{R} - M_e) \,.$$

4. Increment $i = i + 1$ and go to step 2.

We denote stage numbers by $s$ and phase numbers by $i$. We denote by $M_1(i), \ldots, M_r(i)$ the values of the registers $M_1, \ldots, M_r$, respectively, at the end of phase $i$. Similarly, we denote by $N_1(i), \ldots, N_r(i)$ the values of the registers $N_1, \ldots, N_r$, respectively, and by $S_1(i), \ldots, S_r(i)$ the values of the registers $S_1, \ldots, S_r$, respectively, at the end of phase $i$.

In sections 3 and 5, we present performance bounds for the EEE method when the length of the phase is $n = N_e$. In section 6.4 we consider the case where $n = L$ for a fixed $L$. Due to space limitations, proofs are omitted and can be found in the online appendix CITE.

## 3 Bounds Based on Actual Expert Performance

The original variant of the EEE method [2] used $p_i = 1/i$ and $n = N_e$. The following was proven:

$$\Pr\left(\liminf_{s \to \infty} M(s) \geq \max_e \liminf_{i \to \infty} M_e(i)\right) = 1 \,. \tag{1}$$

In words, the algorithm achieves asymptotically an average reward that is as large as that of the best expert. In this section we generalize this result. We present several bounds characterizing the relationship between $M(i)$ and $M_e(i)$. These bounds are valuable in several ways. First, they provide worst-case guarantees about the performance of the EEE method. Second, they provide a starting point for analyzing the behavior of the method under various assumptions about the environment. Third, they quantify the relationship between amount of exploration, represented by the exploration probabilities $p_i$, and the loss of performance. Together with the analysis of Section 5, which characterizes how fast the EEE method learns the value of each expert, the bounds derived here describe explicitly the tradeoff between exploration and exploitation.

We denote by $Z_{ej}$ the event "phase $j$ performs exploration with expert $e$," and let $Z_j = \sum_e Z_{ej}$ and

$$\bar{Z}_{i_0 i} = \mathbf{E}\left[\sum_{j=i_0+1}^{i} Z_j\right] = \sum_{j=i_0+1}^{i} p_i \,.$$

Note that $\bar{Z}_{i_0 i}$ denotes the expected number of exploration phases between phases $i_0 + 1$ and $i$.

The first theorem establishes that, with high probability, after a finite number of iterations, the EEE method performs comparably to the best expert. The performance of each expert is defined as the smallest average reward achieved by that expert in the interval between an (arbitrary) phase $i_0$ and the current phase $i$. It can be shown via a counterexample that this bound cannot be extended into a (somewhat more natural) comparison between the average reward of the EEE method and the average reward of each expert at iteration $i$.

**Theorem 3.1.** *For all $i_0$, $i$ and $\epsilon$ such that $\bar{Z}_{i_0 i} \leq i\epsilon^2/(4\sqrt{r}u^2) - i_0\epsilon/(4u)$,*

$$\Pr\left(M(i) \leq \max_e \min_{i_0+1\leq j\leq i} M_e(j) - 2\epsilon\right) \leq \exp\left\{-\frac{1}{2i}\left(\frac{i\epsilon^2}{4\sqrt{r}u^2} - \frac{i_0\epsilon}{4u} - \bar{Z}_{i_0 i}\right)^2\right\}.$$

The following theorem characterizes the expected difference between the average reward of EEE method and that of the best expert.

**Theorem 3.2.** *For all $i_0 \leq i$ and $\epsilon > 0$,*

$$\mathbf{E}\left[M(i) - \max_e \min_{i_0+1\leq j\leq i} M_e(i)\right] \geq -\epsilon - u\frac{i_0(i_0+1)}{i(i/r+1)} - 2u\left(\frac{3u+2\epsilon}{\epsilon}\right)^2 \frac{\bar{Z}_{i_0 i}}{i}.$$

It follows from Theorem 3.1 that, under certain assumptions on the exploration probabilities, the EEE method performs asymptotically at least as well as the expert that did best. Corollary 3.1 generalizes the asymptotic result established in [2].

**Corollary 3.1.** *If* $\lim_{i\to\infty} \bar{Z}_{0i}/i = 0$*, then*

$$\Pr\left(\liminf_{s\to\infty} M(s) \geq \max_e \liminf_{i\to\infty} M_e(i)\right) = 1. \tag{2}$$

Note that here the average reward obtained by the EEE method is compared with the reward actually achieved by each expert during the same run of the method. It does not have any implication on the behavior of $M_e(i)$, which is analyzed in the next section.

## 4    The Value of an Expert

In this section we analyze the behavior of the average reward $M_e(i)$ that is computed by the EEE method for each expert $e$. This average reward is also used by the method to intuitively estimate the value of expert $e$. So, the question is whether the EEE method is indeed capable of learning the value of the best experts. Thus, we first discuss what is a "learnable value" of an expert. This concept is not trivial especially when the environment is reactive. The obvious definition of a value as the expected average reward the expert could achieve, if followed exclusively, does not work. The previous paper presented an example (see Section 4 in [2]) of a repeated Matching Pennies game, which proved this impossibility. That example shows that an algorithm that attempts to learn what an expert would achieve, if played exclusively, cannot avoid committing fatal "mistakes." In certain environments, every non-trivial learning algorithm must commit such fatal mistakes. Hence, such mistakes cannot, in general, be considered necessarily a weakness of the algorithm. A more realistic concept of value, relative to a certain environment policy $\pi$, is defined as follows, using a real parameter $\tau$.

**Definition 4.1.**

(i) **Achievable $\tau$-Value.** *A real $\mu$ is called an* achievable $\tau$-value *for expert $e$ against an environment policy $\pi$, if there exists a constant $c_\tau \geq 0$ such that, for every stage $s_0$, every possible history $h_{s_0}$ at stage $s_0$ and any number of stages $s$,*

$$\mathbf{E}\left[\frac{1}{s}\sum_{s=s_0+1}^{s_0+s} R(a_e(s), b(s)) \ : \ a_e(s) \sim \sigma_e(h_s), \ b(s) \sim \pi(h_s)\right] \geq \mu - \frac{c_\tau}{s^\tau}.$$

(ii) **$\tau$-Value.** *The $\tau$-value $\mu_e^\tau$ of expert $e$ with respect to $\pi$ is the largest achievable $\tau$-value of $e$:*

$$\mu_e^\tau = \sup\{\ \mu \ : \ \mu \text{ is an achievable } \tau\text{-value}\}. \tag{3}$$

In words, a value $\mu$ is achievable by expert $e$ if the expert can secure an expected average reward during the $s$ stages, between stage $s_0$ and stage $s_0 + s$, which is asymptotically at least as much as $\mu$, regardless of the history of the play prior to stage $s_0$. In [2], we introduced the notion of *flexibility* as a way of reasoning about the value of an expert and when it can be learned. The $\tau$-value can be viewed as a relaxation of the previous assumptions and hence the results here strengthen those of [2]. We note, however, that flexibility does hold when the environment reacts with bounded memory or as a finite automaton.

## 5  Bounds Based on Expected Expert Performance

In this section we characterize how fast the EEE method learns the $\tau$-value of each expert. We can derive the rate at which the average reward achieved by the EEE method approaches the $\tau$-value of the best expert.

**Theorem 5.1.** *Denote* $\bar{\tau} = \min(\tau, 1)$. *For all* $\epsilon > 0$ *and* $i$,

$$\text{if} \quad \frac{4r}{3} \left( \frac{4c_\tau}{\epsilon(2 - \bar{\tau})} \right)^{1/\bar{\tau}} \leq \bar{Z}_{0i} ,$$

$$\text{then} \quad \Pr\left( \inf_{j \geq i} M_e(j) < \mu_e^\tau - \epsilon \right) \leq \frac{33u^2}{\epsilon^2} \exp\left( -\frac{\epsilon^2 \bar{Z}_{0i}}{43u^2 r} \right) .$$

Note from the definition of $\tau$-values that we can only expect the average reward of expert $e$ to be close to $\mu_e^\tau$ if the phase lengths when the expert is chosen are sufficiently large. This is necessary to ensure that the bias term $c_\tau / s^\tau$, present in the definition of the $\tau$-value, is small. The condition on $\bar{Z}_{0i}$ reflects this observation. It ensures that each expert is chosen sufficiently many phases; since phase lengths grow proportionally to the number of phases an expert is chosen, this implies that phase lengths are large enough.

We can combine Theorems 3.1 and 5.1 to provide an overall bound on the difference of the average reward achieved by the EEE method and the $\tau$-value of the best expert.

**Corollary 5.1.** *For all* $\epsilon > 0$, $i_0$ *and* $i$,

$$\text{if} \quad (i) \ \frac{4r}{3} \left( \frac{4c_\tau}{\epsilon(2 - \bar{\tau})} \right)^{1/\bar{\tau}} \leq \bar{Z}_{0i_0} , \quad \text{and} \quad (ii) \ \bar{Z}_{i_0 i} \leq \frac{i\epsilon^2}{4\sqrt{r}u^2} - \frac{i_0 \epsilon}{4u} ,$$

$$\text{then} \quad \Pr\left( M(i) \leq \max_e \mu_e^\tau - 3\epsilon \right) \tag{4}$$

$$\leq \frac{33u^2}{\epsilon^2} \exp\left( -\frac{\epsilon^2 \bar{Z}_{0i_0}}{43u^2 r} \right) + \exp\left\{ -\frac{1}{2i} \left( \frac{i\epsilon^2}{4\sqrt{r}u^2} - \frac{i_0\epsilon}{4u} - \bar{Z}_{i_0 i} \right)^2 \right\} .$$

Corollary 5.1 explicitly quantifies the tradeoff between exploration and exploitation. In particular, one would like to choose $p_j$ such that $\bar{Z}_{0i_0}$ is large enough to make the first term in the bound small, and $\bar{Z}_{i_0 i}$ as small as possible. In Section 6, we analyze several exploration schemes and their effect on the convergence rate of the EEE method.

Here we can also derive from Theorems 3.1 and 5.1 asymptotic guarantees for the EEE method.

**Corollary 5.2.** *If* $\lim_{i \to \infty} \bar{Z}_{0i} = \infty$, *then* $\Pr\left( \liminf_{i \to \infty} M_e(i) \geq \mu_e^\tau \right) = 1$.

The following is an immediate result from Corollaries 3.1 and 5.2:

**Corollary 5.3.** *If* $\lim_{i \to \infty} \bar{Z}_{0i} = \infty$ *and* $\lim_{i \to \infty} \bar{Z}_{0i}/i = 0$, *then*

$$\Pr\left( \liminf_{i \to \infty} M(i) \geq \max_e \mu_e^\tau \right) = 1 .$$

# 6 Exploration Schemes

The results of the previous sections hold under generic choices of the probabilities $p_i$. Here, we discuss how various particular choices affect the speed of exploiting accumulated information, gathering new information and adapting to changes in the environment.

## 6.1 Explore-then-Exploit

One approach to determining exploration schemes is to minimize the upper bound provided in Corollary 5.1. This gives rise to a scheme where the whole exploration takes place before any exploitation. Indeed, according to expression (4), for any fixed number of iterations $i$, it is optimal to let $\bar{Z}_{0i_0} = i_0$ (i.e., $p_j = 1$ for all $j \leq i_0$) and $\bar{Z}_{i_0i} = 0$ (i.e., $p_j = 0$ for all $j > i_0$). Let $U$ denote the upper bound given by (4). It can be shown that the smallest number of phases $i$, such that $U \leq \beta$, is bounded between two polynomials in $1/\epsilon$, $u$, and $r$. Moreover, its dependence on the the total number of experts $r$ is asymptotically $O(r^{1.5})$.

The main drawback of explore-then-exploit is its inability to adapt to changes in the policy of the environment — since the whole exploration occurs first, any change that occurs after exploration has ended cannot be learned. Moreover, the choice of the last exploration phase $i_0$ depends on parameters of the problem that may not be observable. Finally, it requires fixing $\beta$ and $\epsilon$ a priori, and can only achieve optimality within these tolerance parameters.

## 6.2 Polynomially Decreasing Exploration

In [2] asymptotic results were described that were equivalent to Corollaries 3.1 and 5.3 when $p_j = 1/j$. This choice of exploration probabilities satisfies

$$\lim_{i\to\infty} \bar{Z}_{0i} = \infty \quad \text{and} \quad \lim_{i\to\infty} \bar{Z}_{0i}/i = 0 \,,$$

so the corollaries apply. We have, however,

$$\bar{Z}_{0i_0} \leq \log(i_0) + 1 \,.$$

It follows that the total number of phases required for $U$ to hold grows exponentially in $1/\epsilon$, $u$ and $r$. An alternative scheme, leading to polynomial complexity, can be developed by choosing $p_j = j^{-\alpha}$, for some $\alpha \in (0,1)$. In this case,

$$\bar{Z}_{0i_0} \geq \frac{(i_0 + 1)^{1-\alpha}}{1 - \alpha} - 1$$

and

$$\bar{Z}_{0i} \leq \frac{i^{1-\alpha}}{1 - \alpha} \,.$$

It follows that the smallest number of phases that guarantees that $U \leq \beta$ is on the order of

$$i = O\left(\max\left[\frac{u^{\frac{3-\alpha}{1-\alpha}} r^{\frac{3-\alpha}{2(1-\alpha)}}}{\epsilon^{\frac{3-\alpha}{1-\alpha}}}\left(\log\frac{u^2}{\epsilon^2\beta}\right)^{\frac{1}{1-\alpha}}, \frac{u^{\frac{2}{\alpha}} r^{\frac{1}{2\alpha}}}{\epsilon^{\frac{2}{\alpha}}}\right]\right) \,.$$

## 6.3 Constant-Rate Exploration

The previous exploration schemes have the property that the frequency of exploration vanishes as the number of phases grows. This property is required in order to achieve the asymptotic optimality results described in Corollaries 3.1 and 5.3. However, it also makes the EEE method increasingly slower in tracking changes in the policy of the environment. An alternative approach is to use a constant frequency $p_j = \eta \in (0,1)$ of exploration.

Constant-rate exploration does not satisfy the conditions of Corollaries 3.1 and 5.3. However, for any given tolerance level $\epsilon$, the value of $\eta$ can be chosen so that

$$\Pr\left(\liminf_{i\to\infty} M(i) \geq \max_e \mu_e^\tau - \epsilon\right) = 1 \ .$$

Moreover, constant-rate exploration yields complexity results similar to those of the explore-then-exploit scheme. For example, given any tolerance level $\epsilon$, if

$$p_j = \frac{\eta\epsilon^2}{8\sqrt{r}u^2} \qquad (j = 1, 2, \ldots) ;$$

then it follows that $U \leq \beta$ if the number of phases $i$ is on the order of

$$i = O\left(\frac{r^2 u^5}{\epsilon^5} \log \frac{u^2}{\epsilon^2 \beta}\right) \ .$$

## 6.4   Constant Phase Lengths

In all the variants of the EEE method considered so far, the number of stages per phase increases linearly as a function of the number of phases during which the same expert has been followed previously. This growth is used to ensure that, as long as the policy of the environment exhibits some regularity, that regularity is captured by the algorithm. For instance, if that policy is cyclic, then the EEE method correctly learns the long-term value of each expert, regardless of the lengths of the cycles.

For practical purposes, it may be necessary to slow down the growth of phase lengths in order to get some meaningful results in reasonable time. In this section, we consider the possibility of a constant number $L$ of stages in each phase. Following the same steps that we took to prove Theorems 3.1, 3.2 and 5.1, we can derive the following results:

**Theorem 6.1.** *If the EEE method is implemented with phases of fixed length L, then for all $i_0$, $i$, and $\epsilon$, such that*

$$\bar{Z}_{i_0 i} \leq \frac{i\epsilon^2}{2u^2} - \frac{i_0 \epsilon}{2u} \ ,$$

*the following bound holds:*

$$\Pr\left(M(i) \leq \max_e \min_{i_0+1\leq j\leq i} M_e(j) - 2\epsilon\right) \leq \exp\left\{-\frac{1}{2i}\left(\frac{i\epsilon^2}{2u^2} - \frac{i_0\epsilon}{2u} - \bar{Z}_{i_0 i}\right)^2\right\} \ .$$

We can also characterize the expected difference between the average reward of EEE method and that of the best expert.

**Theorem 6.2.** *If the EEE method is implemented with phases of fixed length L, then for all $i_0 \leq i$ and $\epsilon > 0$,*

$$\mathbf{E}\left[M(i) - \max_e \min_{i_0+1\leq j\leq i} M_e(i)\right] \geq -\epsilon - u\frac{i_0}{i} - \frac{2u^2}{\epsilon}\frac{\bar{Z}_{i_0 i}}{i} \ .$$

**Theorem 6.3.** *If the EEE method is implemented with phases of fixed length $L \geq 2$, then for all $\epsilon > 0$,*

$$\Pr\left(\inf_{j\geq i} M_e(j) < \mu_e^\tau - \frac{c_\tau}{L^\tau} - \epsilon\right) \leq \frac{2L^2 u^2}{\epsilon^2} \cdot \exp\left(-\frac{\epsilon^2 \bar{Z}_{0i}}{4L^2 u^2 r}\right) \ .$$

An important qualitative difference between fixed-length phases and increasing-length ones is the absence of the number of experts $r$ in the bound given in Theorem 6.2. This implies that, in the explore-then-exploit or constant-rate exploration schemes, the algorithm requires a number of phases which grows only linearly with $r$ to ensure that

$$\Pr(M(i) \leq \max_e M_e^\tau - c/L^\tau - \epsilon) \leq \beta \ .$$

Note, however, that we cannot ensure performance better than $\max_e \mu_e^\tau - c_\tau/L^\tau$.

# References

[1] Auer, P., Cesa-Bianchi, N., Freund, Y. and Schapire, R.E. (1995) Gambling in a rigged casino: The adversarial multi-armed bandit problem. In *Proc. 36th Annual IEEE Symp. on Foundations of Computer Science*, pp. 322–331, Los Alamitos, CA: IEEE Computer Society Press.

[2] de Farias, D. P. and Megiddo, N. (2004) How to Combine Expert (and Novice) Advice when Actions Impact the Environment. In *Advances in Neural Information Processing Systems 16*, S. Thrun, L. Saul and B. Schölkopf, Eds., Cambridge, MA:MIT Press. `http://books.nips.cc/papers/files/nips16/NIPS2003_CN09.pdf`

[3] Freund, Y. and Schapire, R.E. (1999) Adaptive game playing using multiplicative weights. *Games and Economic Behavior* **29**:79–103.

[4] Littlestone, N. and Warmuth, M.K. (1994) The weighted majority algorithm. *Information and Computation* **108** (2):212–261.
